# A Pylon Model for Semantic Segmentation

**Victor Lempitsky**  **Andrea Vedaldi**  **Andrew Zisserman**

Visual Geometry Group, University of Oxford*

{vilem,vedaldi,az}@robots.ox.ac.uk

## Abstract

Graph cut optimization is one of the standard workhorses of image segmentation since for binary random field representations of the image, it gives globally optimal results and there are efficient polynomial time implementations. Often, the random field is applied over a flat partitioning of the image into non-intersecting elements, such as pixels or super-pixels.

In the paper we show that if, instead of a flat partitioning, the image is represented by a hierarchical segmentation tree, then the resulting energy combining unary and boundary terms can still be optimized using graph cut (with all the corresponding benefits of global optimality and efficiency). As a result of such inference, the image gets partitioned into a set of segments that may come from different layers of the tree.

We apply this formulation, which we call the *pylon model*, to the task of semantic segmentation where the goal is to separate an image into areas belonging to different semantic classes. The experiments highlight the advantage of inference on a segmentation tree (over a flat partitioning) and demonstrate that the optimization in the pylon model is able to flexibly choose the level of segmentation across the image. Overall, the proposed system has superior segmentation accuracy on several datasets (Graz-02, Stanford background) compared to previously suggested approaches.

## 1 Introduction

Semantic segmentation (i.e. the task of assigning each pixel of a photograph to a semantic class label) is often tackled via a "flat" conditional random field model [10, 29]. This model considers the subdivision of an image into small non-overlapping elements (pixels or small superpixels). It then learns and evaluates the likelihood of each element as belonging to one of the semantic classes (*unary terms*) and combine these likelihoods with *pairwise terms* that encourage neighboring elements to take the same labels, and in this way propagates the information from elements that are certain about their labels to uncertain ones. The appeal of the flat CRF model is the availability of efficient MAP inference based on graph cut [7], which is exact for two-label problems with submodular pairwise terms [4, 16] and gets very close to global optima for many practical cases of multi-label segmentation [31].

The main limitation of the flat CRF model is that since each superpixel takes only one semantic label, super-pixels have to be small, so that they do not straddle class boundaries too often. Thus, the amount of visual information inside the superpixel is limited. The best performing CRF models therefore consider wider local *context* around each superpixel, but as the object and class boundaries are not known in advance, the support area over which such context information is aggregated is not adapted. For this reason, such context-based descriptors have limited repeatability and may not allow reliable classification. This is, in fact, a manifestation of a well-known chicken-and-egg problem between segmentation and recognition (given spatial support based on proper segmentation, recognition is easy [20], but to get the proper segmentation prior recognition is needed).

Recently, several semantic segmentation methods that explicitly interleave segmentation and recognition have been proposed. Such methods [8, 11, 18] consider a large *pool* of overlapping segments that are much bigger

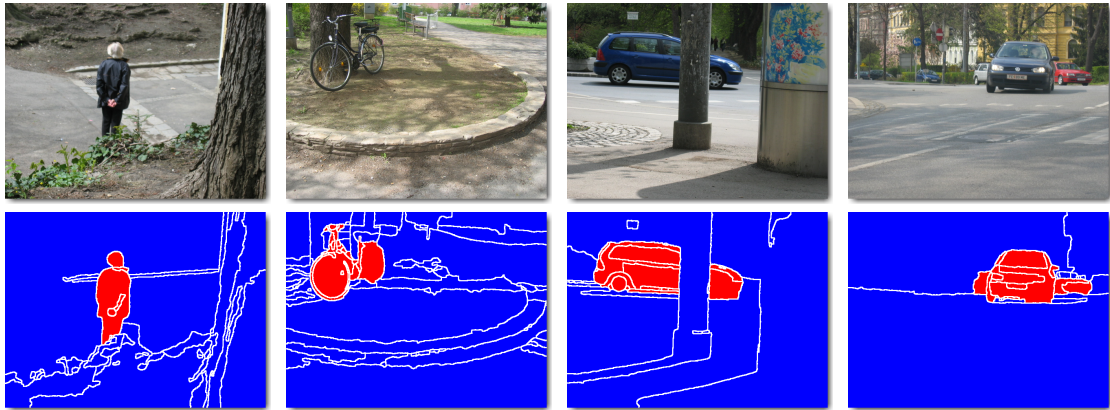

Figure 1: **Pool-based binary segmentation.** For binary semantic segmentation, the pylon model is able to find a globally optimal subset of segments and their labels (bottom row), while optimizing unary and boundary costs. Here we show a result of such inference for images from each of the Graz-02 [23] datasets (people and bikes – left, cars – right).

than superpixels in flat CRF approaches. These methods then perform joint optimization over the choice of several non-overlapping segments from the pool and the semantic labels of the chosen segments. As a result, in the ideal case, a photograph is pieced from a limited number of large segments, each of which can be unambiguously assigned to one of the semantic classes, based on the information contained in it. Essentially, the photograph is then "explained" by these segments that often correspond to objects or their parts. Such scene explanation can then be used as a basis for more high-level scene understanding than just semantic segmentation.

In this work, we present a *pylon model* for semantic segmentation which largely follows the pool-based semantic segmentation approach from [8, 11, 18]. Our goal is to overcome the main problem of existing pool-based approaches, which is the fact that they all face very hard optimization problems and tackle them with rather inexact and slow algorithms (greedy local moves for [11], loose LP relaxations in [8, 18]). Our aim is to integrate the exact and efficient inference employed by flat CRF methods with the strong scene interpretation properties of the pool-based approaches.

Like previous pool-based approaches, the pylon model "explains" each image as a union of non-intersecting segments. We achieve the tractability of the inference by restricting the pool of segments to come from a *segmentation tree*. Segmentation trees have been investigated for a long time, and several efficient algorithms have been developed [1, 2, 38, 27]. Furthermore, any binary unsupervised algorithm (e.g. normalized cut [28]) can be used to obtain a segmentation tree via iterative application. As segmentation trees reflect the hierarchical nature of visual scenes, algorithms based on segmentation-trees achieved very impressive results for visual-recognition tasks [13, 22, 34]. For our purpose, the important property of tree-based segment pool is that each image region is covered by segments of very different sizes and there is a good chance that one such segment does not straddle object boundaries but is still big enough to contain enough visual information for a reliable class identification.

Inference in pylons optimizes the sum of the real-valued costs of the segments selected to explain the image. Similarly to random field approaches, pylons also include spatial smoothness terms that encourage the boundary compactness of the resulting segmentations (this could be e.g. the popular contrast-dependent Potts-potentials). Such boundary terms often remedy the imperfections of segmentation trees by propagating the information from big segments that fit within object boundaries to smaller ones that have to supplement the big segments to fit class boundaries accurately.

The most important advantage of pylons over previous pool-based methods [8, 11, 18] is the tractability of inference. Similarly to flat CRFs, in the two-class (e.g. foreground-background) case, the globally optimal set of segments can be found exactly and efficiently via graph cut (Figure 1). Such inference can then be extended to multi-label problems via an alpha-expansion procedure [7] that gives solutions close to a global optimum. Effectively, inference in pylons is as "easy" as in the flat CRF approach. We then utilize such a "free lunch" to achieve a better than state-of-the-art performance on several datasets (Graz-02 datasets[23] for binary label segmentations, Stanford background dataset [11] for multi-label segmentation). At least in part, the excellent performance of our system is explained by the fact that we can learn both unary and boundary term parameters within a standard max-margin approach developed for CRFs [32, 33, 35], which is

not easily achievable with the approximate and slow inference in previous pool-based methods [17]. We also demonstrate that the pylon model achieves higher segmentation accuracy than flat CRFs, or non-loopy pylon models without boundary terms, given the same features and the same learning procedure.

**Other related work.** The use of segmentation trees for semantic segmentation has a long history. The older works of [5] and [9] as well as a recent work [22] use a sequence of top-down inference processes on a segmentation tree to infer the class labels at the leaf level. Our work is probably more related to the approaches performing MAP estimation in tree-structured/hierarchical random fields. For this, Awasthi et al. [3], Reynolds and Murphy [25] and Plath et al. [24] use pure tree-based random fields without boundary terms, while Schnitzspan et al. [26] and Ladicky et al. [19] incorporate boundary terms and perform semantic segmentation at different levels of granularity. The weak consistency between levels is then enforced with higher-order potentials. Overall, our philosophy is different from all these works as we obtain an explicit scene interpretation as a union of few non-intersecting segments, while the tree-structured/hierarchical CRF works assign class labels and aggregate unary terms over all segments in the tree/hierarchy. Our inference however is similar to that of [19]. In fact, while below we demonstrate how inference in pylons can be reduced to submodular pseudo-boolean quadratic optimization, it can also be reduced to the hierarchical associative CRFs introduced in [19]. We also note that another interesting approach to joint segmentation and classification based on this class of CRFs has been recently proposed by Singaraju and Vidal [30].

## 2    Random fields, Pool-based models, and Pylons

We now derive a joint framework covering the flat random field models, the preceding pool-based models, and the pylon model introduced in this paper.

We consider a semantic segmentation problem for an image $I$ and a set of $K$ semantic classes, so that each part of the image domain has to be assigned to one of the classes. Let $\mathbf{S} = \{S_i | i = 1 \ldots N\}$ be a pool of segments, i.e. a set of sub-regions of the image domain. For a traditional (flat) random field approach, this pool comes from an image partitioned into is a set of small non-intersecting segments (or pixels); in the case of the pool-based models this is an arbitrary set of many segments coming from multiple flat segmentations [18] or explored via local moves [11]. In the pylon case, $\mathbf{S}$ contains all segments in a segmentation tree computed for an image $I$.

A segmentation $\mathbf{f}$ then assigns each $S_i$ an integer label $f_i$ within a range from 0 to $K$. A special label $f_i{=}0$ means that the segment is not included into the segmentation, while the rest of the labels mean that the segment participates in the explanation of the scene and is assigned to a semantic class $f_i$. Not all labelings are consistent and correspond to valid segmentations. First of all, the *completeness* constraint requires that each image pixel $p$ is covered by a segment with non-zero label:

$$\forall p \in I, \ \exists i: \ S_i \ni p, f_i > 0 \tag{1}$$

For the flat random field case, this means that zero labels are prohibited and each segment has to be assigned some non-zero label. For pool-based methods and the pylon model, this is not the case as each pixels has a multitude of segments in $\mathbf{S}$ covering it. Thus, zero labels are allowed. Furthermore, non-zero labels should be controlled by the *non-overlap* constraint requiring that overlapping segments cannot take non-zero labels:

$$\forall i \neq j: \ S_i \cap S_j \neq \emptyset \Rightarrow f_i \cdot f_j = 0 \ . \tag{2}$$

Once again, the constraint (2) is not needed for flat CRFs as their pools do not contain overlapping segments. It is, however, non-trivial for the existing pool-based models and for the pylon model, where overlapping (nested) segments exist. Under the constraints (1) and (2), each pixel $p$ in the image is covered by exactly one segment with non-zero label and we define the number of this segment as $i(p)$. The semantic label $f(p)$ of the pixel $p$ is then determined as $f_{i(p)}$.

To formulate the energy function, we define the set of real-valued unary terms $U_i(f_i)$, where each $U_i$ specifies the cost of including a segment $S_i$ into the segmentation with the label $f_i > 0$. Furthermore, we associate the non-negative boundary cost $V_{pq}$ with any pair of pixels adjacent in the image domain $(p, q) \in \mathcal{N}$. For any segmentation $\mathbf{f}$ we then define the *boundary cost* as the sum of boundary costs over the sets of adjacent pixel pairs $(p, q)$ that straddle the boundaries between classes induced by this segmentation (i.e. $(p, q) \in \mathcal{N}: f(p) \neq f(q)$). In other words, the boundary terms are accumulated along the boundary between pool segments that are assigned different non-zero semantic labels.

Overall, the energy that we are interested in, is defined as:

$$E(\mathbf{f}) = \sum_{i \in 1..N | f_i > 0} U_i(f_i) + \sum_{(p,q) \in \mathcal{N}: f(p) \neq f(q)} V_{pq} \tag{3}$$

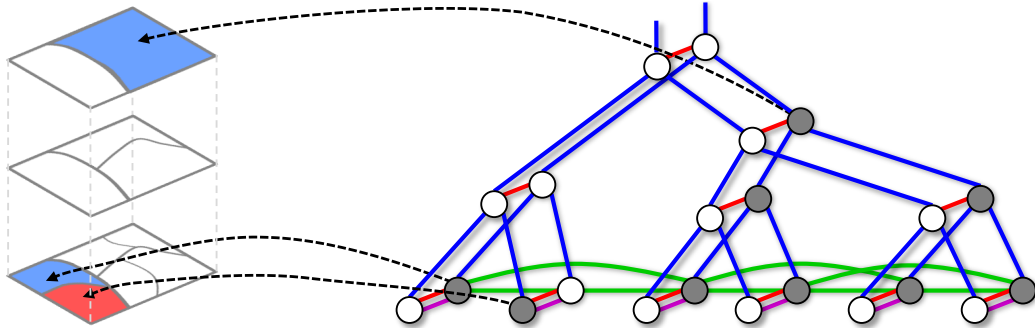

Figure 2: **Inference in the Pylon model***(best viewed in color.)*: a tree segmentation of an image (left) and a corresponding graphical model for the 2-class pylon (right). Each pair of nodes in the graphical model correspond to a segment in a segmentation tree, while each edge corresponds to the pairwise term in the pseudo-boolean energy (9)–(10). Blue edges (4) enforce the segment cost potentials ($U$-terms) as well as consistency of **x** (children of a shaded node have to be shaded). Red edges (6) and magenta edges (7) enforce non-overlap and completeness. Green edges (8) encode boundary terms. Shading gives an example valid labeling for **x** variables ($x_i^t$=1 are shaded). Left – the corresponding semantic segmentation on the segmentation tree consisting of three segments is highlighted.

and we wish to minimize this subject to the constraints (1) and (2). The energy (3) contains the contribution of unary terms only from those segments that are selected to explain the image ($f_i > 0$).

Note that the energy functional has the same form as that of a traditional random field (with weighted Potts boundary terms). The pool-based model in [18] is also similar, but lacks the boundary terms. It is well-known that for flat random fields, the optimal segmentation **f** in the binary case $K = 2$ with $V_{pq} \geq 0$ can be found with graph cut [7, 12, 16]. Furthermore, for $K > 2$ one can get very close to global optimum (within a factor 2 with guarantee [7], but much closer in practice [31]) by applying graph cut-based alpha-expansions [7].

For pylons as well as for the pool-based approaches [11, 18], the segment pool is much richer. As a consequence, the constraints (1) and (2) that are trivial to enforce in the case of the flat random field, become non-trivial. In the next section, we demonstrate that in the case of a tree-based pool of segments (pylon model), one still can find the globally optimal **f** in the case $K = 2$ and $V_{pq} \geq 0$, and use alpha-expansions in the case $K > 2$.

**1-class model.** Before discussing the inference and learning in the pylon model, we briefly introduce a modification of the generic model derived above, which we call a *1-class model*. A 1-class model can be used for semantic foreground-background segmentation tasks (e.g. segmenting out people in an image). The 2-class model defined in (1)–(3) for $K = 2$ can of course also be used for this purpose. The difference is that the 1-class model treats the foreground and background in an asymmetric way. Namely, for 1-class case the labels $x_i$ can only take the values of 0 or 1 (i.e. $K$=1) and the completeness constraint (1) is omitted. As such, each segmentation **f** defines the foreground as a set of segments with $f_i$=1 and the semantic label of a pixel $f(p)$ is defined to be 1 if $p$ belongs to some segment $S_i$ with $f_i = 1$ and $f(p) = 0$ otherwise. In a 1-class case, each segment has thus a single unary cost $U_i = U_i(1)$ associated with it. The energy remains the same as in (3).

For the flat random field case, the 1-class and 2-class models are equivalent (one can just define $U_i^{\text{1class}} = U_i^{\text{2class}}(2) - U_i^{\text{2class}}(1)$ to get the same energy upto an additive constant). For pool-based models and pylons, this is no longer the case, and the 1-class model is non-trivially different from the 2-class model. Intuitively, a 1-class model only "explains" the foreground as a union of segments, while leaving the background part "unexplained". As shown in our experiments, this may be beneficial e.g. when the visual appearance of foreground is more repeatable than that of the background.

## 3 Inference in pylon models

**Two-class case.** We first demonstrate how the energy (3) can be globally minimized subject to (1)–(2) in the case of a tree-based pool and $K = 2$. Later, we will outline inference in the case $K > 2$ and in the case of a 1-class model $K = 1$. For each segment number $i = 1..N$ we define $p(i)$ to be the number of its parent segment in a tree. We further assume that the first $L$ segments correspond to leaves in the segmentation tree and that the last segment $S_N$ is the root (i.e. the entire image).

For each segment $i$, we introduce two binary variables $x_i^1$ and $x_i^2$ indicating whether the segment falls entirely into the segment assigned to class 1 or 2. The exact semantic meaning and relation to variables $\mathbf{f}$ of these labels is as follows: $x_i^t$ equals 1 if and only if one of its ancestors $j$ up the tree (including the segment $i$ itself) has a label $f_j = t$. We now re-express the constraints (1)–(2) and the energy (3) via a real valued (i.e. *pseudo-boolean*) energy of the newly-introduced variables that involve pairwise terms only (Figure 2).

First of all, the definition of the $\mathbf{x}$ variables implies that if $x_i^t$ is zero, then $x_{p(i)}^t$ has to be zero as well. Furthermore, if $x_i^t = 1$ and $x_{p(i)}^t = 0$ implies that the segment $i$ has a label $f_i = t$ (incurring the cost $U_i(t)$ in (3)). These two conditions can be expressed with the bottom-up pairwise term on the variables $x_i^t$ and $x_{p(i)}^t$ (one term for each $t = 1, 2$):

$$E_i^t(0,0) = 0, \ E_i^t(0,1) = +\infty, \quad E_i^t(1,0) = U_i(t), \ E_i^t(1,1) = 0\,. \tag{4}$$

These potentials express almost all unary terms in (3) except for the unary term for the root node, that can be expressed as a sum of two unary terms on the new variables (one term for each $t = 1, 2$):

$$E_N^t(0) = 0, \quad E_N^t(1) = U_N(t)\,. \tag{5}$$

The non-overlap constraint (2) can be enforced by demanding that at most one of $x_i^1$ and $x_i^2$ can be 1 at the same time (as otherwise there are two segments with non-zero $f$-variables that overlap), introducing the following *exclusion* pairwise term on the variables $x_i^1$ and $x_i^2$:

$$E_i^{EXC}(0,0) = E_i^{EXC}(0,1) = E_i^{EXC}(1,0) = 0, \quad E_i^{EXC}(1,1) = +\infty\,. \tag{6}$$

The completeness constraint (1) can be expressed by demanding that each leaf segment is covered by either an ancestor segment with label 1 or with label 2. Consequently, in the leaf node, at least one of $x_i^1$ and $x_i^2$ has to be 1, hence the following pairwise *completeness* potential for all leaf segments $i = 1..L$:

$$E_i^{CPL}(0,0) = +\infty, \quad E_i^{CPL}(0,1) = E_i^{CPL}(1,0) = E_i^{CPL}(1,1) = 0\,. \tag{7}$$

Finally, the only unexpressed part of the optimization problem is the boundary term in (3). To express the boundary term, we consider the set $\mathcal{P}$ of pairs of numbers of adjacent leaf segments. For each such pair $(i, j)$ of leaf segments $(S_i, S_j)$ we consider all pairs of adjacent pixels $(p, q)$. The boundary cost $V_{ij}$ between $S_i$ and $S_j$ is then defined as the sum of pixel-level pairwise costs $V_{ij} = \sum V_{pq}$ over all pairs of adjacent pixels $(p, q) \in \mathcal{N}$ such that $p \in S_i$ and $q \in S_j$ or vice versa (i.e. $p \in S_j$ and $q \in S_i$). The boundary terms can then be expressed with pairwise terms over variables $x_i^1$ and $x_j^1$ for all $(i, j) \in \mathcal{P}$:

$$E_{ij}^{BND}(0,0) = E_{ij}^{BND}(1,1) = 0, \quad E_{ij}^{BND}(0,1) = E_{ij}^{BND}(1,0) = V_{ij}\,. \tag{8}$$

Overall, the constrained minimization problem (1)–(3) for the variables $\mathbf{f}$, is expressed as the unconstrained minimization of the following energy of boolean variables $\mathbf{x}^1, \mathbf{x}^2$:

$$E(\mathbf{x}^1, \mathbf{x}^2) = \sum_{t=1,2} \sum_{i=1}^{N-1} E_i^t(x_i^t, x_{p(i)}^t) + \sum_{t=1,2} E_N^t(x_N^t) + \sum_{(i,j)\in\mathcal{P}} E_{i,j}^{BND}(x_i^1, x_j^1) + \tag{9}$$

$$\sum_{i=1}^{N} E_i^{EXC}(x_i^1, x_i^2) + \sum_{i=1}^{L} E_i^{CPL}(x_i^1, x_i^2) \tag{10}$$

The energy (9)–(10) contains two parts. The pairwise terms in the first part (9) involve only such pairs of variables that both terms come either from $\mathbf{x}^1$ set or from $\mathbf{x}^2$ set. All the pairwise terms in (9) are *submodular*, i.e. they obey $E(0,0) + E(1,1) \leq E(0,1) + E(1,0)$. The pairwise terms in the second part (10) involve only such pairs of variables where one term comes from the $\mathbf{x}^1$ set and the other from the $\mathbf{x}^2$ set. All terms in (10) are *supermodular*, i.e. obey $E(0,0) + E(1,1) \geq E(0,1) + E(1,0)$.

Thus, in the energy (9)–(10), submodular terms act within $\mathbf{x}^1$ and $\mathbf{x}^2$ sets of variables and supermodular terms act only across the two sets. One can then perform a variable substitution $\mathbf{x}^2 = 1 - \tilde{\mathbf{x}}^2$, and get a new energy function $E(\mathbf{x}^1, \tilde{\mathbf{x}}^2)$. During the substitution, the terms (9) remain submodular, while the terms (10) change from being supermodular to being submodular in the new variables. As a result, one gets a pseudo-boolean pairwise energy with submodular terms only, which can therefore be minimized exactly and in a low-polynomial in $N$ time through the graph cut in a specially constructed graph [4, 6, 16]. Given the

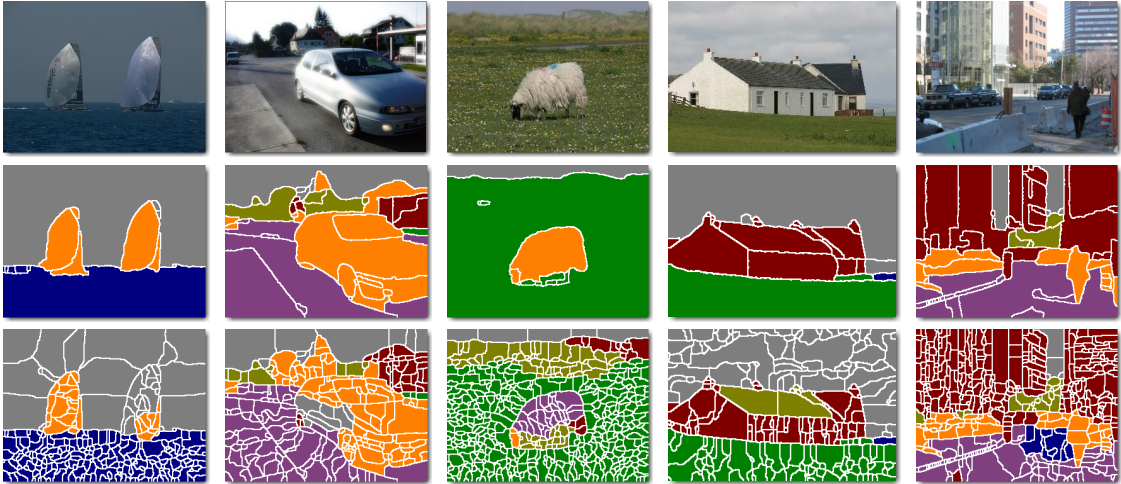

Figure 3: Several examples from the Stanford background dataset [11], where the ability of the pylon model (middle row) to choose big enough segments allowed it to obtain better semantic segmentation compared to a flat CRF defined on leaf segments (bottom row). Colors: *grey=sky, olive=tree, purple=road, green=grass, blue=water, red=building, orange=foreground.*

optimal values for $\mathbf{x}^1$ and $\tilde{\mathbf{x}}^2$, it is trivial to infer the optimal values for $\mathbf{x}^2$ and ultimately for the $\mathbf{f}$ variables (for the latter step one goes up the tree and set $f_i = t$ whenever $x_i^t = 1$ and $x_{p(i)}^t = 0$).

**One-class case.** Inference in the one-class case is simpler that in the two-class case. As one may expect, it is sufficient to introduce just a single set of binary variables $\{x_1^i\}$ and omit the pairwise terms (6) and (7) altogether. The resulting energy function is then:

$$E(\mathbf{x}^1) = \sum_{i=1}^{N-1} E_i^1(x_i^1, x_{p(i)}^1) + E_N^1(x_N^1) + \sum_{(i,j)\in\mathcal{P}} E_{i,j}^{BND}(x_i^1, x_j^1) \qquad (11)$$

In this case, the non-overlap constraint is enforced by infinite terms within (4). The pseudo-boolean energy (11) is submodular and, hence, can be optimized directly via graph cut.

**Multi-class case.** As in the flat CRF case, the alpha-expansion procedure [7] can be used to extend the 2-class inference procedure to the case $K > 2$. Alpha-expansion is an iterative convergent process, where 2-class inference is applied at each iteration. In our case, given the current labeling $\mathbf{f}$, and a particular $\alpha \in 1 \dots K$, each segment has the following three options: (1a) a segment with the non-zero label can retain it (1b) a segment with zero label can change it to the current non-zero label of its ancestor (if any), (2) label $f_i$ can be changed to $\alpha$, (3) label $f_i$ can be changed to $0$ (or kept at $0$ if already there). Thus, each step results in the 2-class inference task, where $U$ and $V$ potentials of the 2-class inference are induced by the $U$ and $V$ potentials of the multi-label problem (in fact, some boundary terms then become asymmetric if one of the adjacent segments have the current label $\alpha$. We do not detail this case here since it is handled in exactly the same way as in [7]). Alpha-expansion then performs a series of 2-class inferences for $\alpha$ sweeping the range $1 \dots K$ multiple times until convergence.

## 4 Implementation and Experiments

**Segmentation tree.** For this paper, we used the popular segmentation tree approach [2] that is based on the powerful *pPb* edge detector and is known to produce high-quality segmentation trees. The implementation [2] is rather slow (orders of magnitude slower than our inference) and we plan to explore faster segmentation tree approaches.

**Features.** We use the following features to describe a segment $S_i$: (1) a histogram $h_i^{\text{SIFT}}$ of densely sampled visual SIFT words computed with `vl_feat` [36]. We use a codebook of size 512, and soft-assign each word to the 5 nearest codewords via the locality-constrained linear coding [39]; (2) a histogram $h_i^{\text{COL}}$ of RGB colors (codebook size 128; hard-assignment); (3) a histogram $h_i^{\text{LOC}}$ of locations (where each pixel corresponds to a number from 1 to 36 depending on its position in a uniform $6 \times 6$ grid; (4) the "contour shape" descriptor $h_i^{\text{SHP}}$ from [13] (a binned histogram of oriented *pPb* edge detector responses). Each of the four histograms is

then normalized and mapped by a non-linear coordinate-wise mapping $H(\cdot)$ to a higher-dimensional space, where the inner product (linear kernel) closely approximates the $\chi^2$-kernel in the original space [37]. The unary term $U_i^t$ is then computed as a scalar product of the stacked descriptor and the parameter weight vector $w_U^t$:

$$U_i^t = s_i \cdot \left[ H(h_i^{\text{SIFT}})^T \quad H(h_i^{\text{COL}})^T \quad H(h_i^{\text{LOC}})^T \quad H(h_i^{\text{SHP}})^T \quad 1 \right] \cdot w_U^t . \tag{12}$$

Note, that each unary term is also multiplied by $s_i$, which is the size of the segment $S_i$. Without such multiplication, the inference process would be biased towards small segments (leaves in the segmentation trees).

The boundary cost for a pair of pixel $(p, q) \in \mathcal{N}$ is set based on the local boundary strength $\Delta_{pq}$ estimated with *gPb* edge detector. The exact value of $V_{pq}$ is then computed as a linear combination of exponentiated $\Delta_{pq}$ with several bandwidths:

$$V_{pq} = \left[ \exp\left( \frac{-\Delta_{pq}}{10} \right) \quad \exp\left( \frac{-\Delta_{pq}}{40} \right) \quad \exp\left( \frac{-\Delta_{pq}}{100} \right) \quad 1 \right] \cdot w_V \tag{13}$$

We discuss the learning of parameters $w$ below. The meta-parameters (codebook sizes, number of words in soft-assignment, number of bins for location and contour shape descriptors, bandwidths) were not tweaked (we set them based on previous experience and have not tried other values).

**Max-margin learning parameters.** Denote by $\mathbf{w} = [w_U^1, \dots, w_U^K, w_V]$, $w_V \geq 0$ the parameter of the pylon model $(\hat{\mathbf{x}}^1(\mathbf{w}), \hat{\mathbf{x}}^2(\mathbf{w}))$, defined as the minimizer of the energy $E(\mathbf{x}^1, \mathbf{x}^2)$ given in (9)–(10). The goal is to find a parameter $\mathbf{w}$ such that $(\hat{\mathbf{x}}^1(\mathbf{w}), \hat{\mathbf{x}}^2(\mathbf{w}))$ has a small Hamming distance $\Delta(\hat{\mathbf{x}}^1(\mathbf{w}), \hat{\mathbf{x}}^2(\mathbf{w}))$ to the segmentation $\bar{\mathbf{x}}^1, \bar{\mathbf{x}}^2$ of a training image. The Hamming distance is simply the number of pixels incorrectly labeled. To obtain a convex optimization problem and regularize its solution, we use the large margin formulation of [33, 14]. The first step is to rewrite the optimization task (9)–(10) as:

$$(\hat{\mathbf{x}}^1(\mathbf{w}), \hat{\mathbf{x}}^2(\mathbf{w})) = \underset{\mathbf{x}^1, \mathbf{x}^2}{\text{argmax}} -E(\mathbf{x}^1, \mathbf{x}^2) = \underset{\mathbf{x}^1, \mathbf{x}^2}{\text{argmax}} F(\mathbf{x}^1, \mathbf{x}^2) + \langle \Psi(\mathbf{x}^1, \mathbf{x}^2), \mathbf{w} \rangle, \tag{14}$$

where $\Psi(\mathbf{x}^1, \mathbf{x}^2)$ is a concatenation of the summed coefficients of (12) and (13) and $F(\mathbf{x}^1, \mathbf{x}^2)$ accounts for the terms of $E(\mathbf{x}^1, \mathbf{x}^2)$ that do not depend on $\mathbf{w}$. Then margin rescaling [14] is used to construct a convex upper bound of the Hamming loss $\Delta(\hat{\mathbf{x}}^1(\mathbf{w}), \hat{\mathbf{x}}^2(\mathbf{w}))$:

$$\Delta'(\mathbf{w}) = \underset{\mathbf{x}^1, \mathbf{x}^2}{\max} \Delta(\mathbf{x}^1, \mathbf{x}^2) + F(\mathbf{x}^1, \mathbf{x}^2) - F(\bar{\mathbf{x}}^1, \bar{\mathbf{x}}^2) + \langle \Psi(\mathbf{x}^1, \mathbf{x}^2), \mathbf{w} \rangle - \langle \Psi(\bar{\mathbf{x}}^1, \bar{\mathbf{x}}^2), \mathbf{w} \rangle \tag{15}$$

The function $\Delta'(\mathbf{w})$ is convex because it is the upper envelope of a family of planes, one for each setting of $\mathbf{x}^1, \mathbf{x}^2$. This allows to learn the parameter $\mathbf{w}$ as the minimizer of the convex objective function $\lambda \|\mathbf{w}\|^2 / 2 + \Delta'(\mathbf{w})$, where $\lambda$ controls overfitting. Optimization uses the cutting plane algorithm described in [14], which gradually approximates $\Delta'(\mathbf{w})$ by selecting a small representative subset of the exponential number of planes that figure in (15). These representative planes are found by maximizing (15), which can be done by the algorithm described in Sect. 3 after accounting for the loss $\Delta(\mathbf{x}^1, \mathbf{x}^2)$ in a suitable adjustment of the potentials.

**Datasets.** We consider the three **Graz-02** datasets [23] that to the best of our knowledge represent the most challenging datasets for semantic binary (foreground-background) segmentation. Each Graz-02 dataset has one class of interest (bicycles, cars, and people). The datasets are loosely annotated at the pixel level. Previous methods reported performance for the fixed splits including 150 training and 150 testing images. The customary performance measure is the equal recall-precision rate averaged over all pixels in the test set. In general, when trained with Hamming loss, our method produces recall slightly lower than precision. We therefore retrained our system with weighted Hamming loss (so that false negatives are penalized higher than false positive), tuning the balancing constant to achieve approximately equal recall and precision (an alternative would be to use parametric maxflow [15]).

We also consider the **Stanford background** dataset [11] containing 715 images of outdoor scenes with pixel-accurate annotations into 8 semantic classes (sky, tree, road, grass, water, building, mountain, and foreground object). Similar to previous approaches, we report the percentage of correctly labeled pixels on 5 random splits of a fixed size (572 training, 143 testing).

**Results.** We compare the performance of our system with the state-of-the-art in Table 1. We note that our approach performs considerably better than state-of-the-art including the CRF-based method [10], the pool-based methods [11, 18], and the approach based on the same *gPb*-based tree [22]. There are probably three

| Graz-02 dataset [23] | equal recall-precision | | |
|---|---|---|---|
| Method | Bikes | Cars | People |
| Marszalek&Schmid [21] | 53.8 | 44.1 | 61.8 |
| Fulkerson et al. [10] | 72.2 | 72.2 | 66.3 |
| **1-class pylon** | 83.4 | **84.9** | 81.5 |
| **2-class pylon** | **83.7** | 83.3 | **82.5** |

| Stanford background dataset [11] | |
|---|---|
| Method | correct % |
| Gould et al. [11] | 76.4 ± 1.22 |
| Munoz et al. [22] | 76.9 |
| Kumar&Koller [18] | 79.42 ± 1.41 |
| **8-class pylon** | **81.90 ± 1.09** |

Table 1: Comparison with state-of-the-art. Left – equal recall-precision on the Graz datasets (pylon models were trained with class-weighted Hamming loss to achieve approximately equal recall-precision). Right – percentage of correctly labelled pixels on the Stanford dataset. For all datasets, our systems achieves a considerable improvement over the state-of-the-art.

| Model | Graz-02 Bikes | | | Graz-02 Cars | | | Graz-02 People | | | Stanford background | |
|---|---|---|---|---|---|---|---|---|---|---|---|
| | rec. | prec. | Ham. | rec. | prec. | Ham. | rec. | prec. | Ham. | mean | diff. to full |
| 1-class pylon | **80.8** | **86.9** | **7.7** | **81.7** | **87.0** | **3.1** | 77.3 | 85.0 | 6.4 | – | – |
| 2/8-class pylon | 81.2 | 86.1 | 7.8 | 80.4 | 85.6 | 3.4 | **78.7** | **84.4** | **6.3** | 81.90 | **0.00 ± 0.00** |
| Flat CRF – 0 | 79.4 | 83.8 | 8.8 | 80.7 | 86.8 | 3.3 | 73.7 | 79.8 | 7.9 | 80.07 | −1.84 ± 0.15 |
| Flat CRF – 20 | 81.3 | 84.6 | 8.2 | 81.1 | 83.7 | 3.6 | 76.7 | 80.7 | 7.3 | 81.13 | −0.78 ± 0.42 |
| Flat CRF – 40 | 78.3 | 85.4 | 8.6 | 81.2 | 82.1 | 3.8 | 76.0 | 80.6 | 7.4 | 80.25 | −1.65 ± 0.69 |
| Flat CRF – 60 | 71.2 | 84.2 | 10.3 | 79.5 | 80.8 | 4.1 | 71.6 | 79.0 | 8.4 | 77.99 | −3.91 ± 0.74 |
| Flat CRF – 80 | 64.5 | 81.1 | 12.4 | 74.7 | 76.8 | 4.9 | 68.9 | 80.2 | 8.4 | 75.01 | −6.89 ± 0.47 |
| 1-class pylon (no bnd) | 78.3 | 85.7 | 8.5 | 76.7 | 83.9 | 3.9 | 76.3 | 84.9 | 6.6 | – | – |
| 2/8-class pylon (no bnd) | 79.6 | 85.7 | 8.3 | 77.9 | 84.0 | 3.8 | 76.6 | 82.9 | 6.9 | 81.29 | −0.62 ± 0.24 |

Table 2: Comparison with baseline methods with the same features and the same training procedure (unweighted Hamming loss was used in all cases). 'Flat CRF – X' correspond to flat random fields trained and evaluated on the segmentations obtained by thresholding the segmentation tree at level X. The last two lines correspond to the pylon model trained and evaluated with boundary terms disabled. For Graz-02, recall, precision and Hamming error for the predefined splits are given. For Stanford background, % of correctly-labeled pixels is measured over 5 random splits, then the mean and the difference to the full pylon model are given. For all datasets, the full pylon models perform better than the baselines (the best baseline for each dataset is underlined).

reasons for this higher performance: superior features, a superior learning procedure, and a superior model (pylon).

To clarify what is the benefit of the pylon model alone, we perform an extensive comparison with baselines (Table 2). We compare with the flat CRF approaches, where the partitions are obtained by thresholding the segmentation tree at different levels. We also determine the benefit of having boundary terms by comparing with the pylon model without these terms. All baseline models used the same features and the same max-margin learning procedure. The full pylon model performs better than baselines, although the advantage is not as large as that over the preceding methods.

**Efficiency.** The runtime of the entire framework is dominated by the pre-computation of segmentation trees and the features. After such pre-computation, our graph cut inference is extremely fast: less than 0.1s per image/label which is orders of magnitude faster than inference in previous pool-based methods. Training the model (after the precomputation) takes 85 minutes for one split of the Stanford background dataset (compared to 55 minutes for the flat CRF).

## 5 Discussion

Despite a very strong performance of our system in the experiments, we believe that the main appeal of the pylon model is in the combination of interpretability, tractability, and flexibility. The interpretability is not adequately measured by the quantitative evaluation, but it may be observed in qualitative examples (Figures 1 and 3), where many segments chosen by the pylon model to "explain" a photograph correspond to objects or their high-level parts. The pylon model generalizes the flat CRF model for semantic segmentation that operates with small low-level structural elements. Notably, despite such generalization, the inference and max-margin learning in the pylon model is as easy as in the flat CRF model.

## Footnotes

*Victor Lempitsky is currently with Yandex, Moscow. This work was supported by ERC grant VisRec no. 228180 and by the PASCAL Network of Excellence.

# References

[1] N. Ahuja. A transform for multiscale image segmentation by integrated edge and region detection. *IEEE Trans. Pattern Anal. Mach. Intell.*, 18(12), 1996.

[2] P. Arbelaez, M. Maire, C. Fowlkes, and J. Malik. Contour detection and hierarchical image segmentation. *IEEE Trans. Pattern Anal. Mach. Intell.*, 33(5):898–916, 2011.

[3] P. Awasthi, A. Gagrani, and B. Ravindran. Image modeling using tree structured conditional random fields. In *IJCAI*, pages 2060–2065, 2007.

[4] E. Boros and P. L. Hammer. Pseudo-boolean optimization. *Discrete Applied Mathematics*, 123(1-3):155–225, 2002.

[5] C. A. Bouman and M. Shapiro. A multiscale random field model for bayesian image segmentation. *IEEE Transactions on Image Processing*, 3(2):162–177, 1994.

[6] Y. Boykov and V. Kolmogorov. An experimental comparison of min-cut/max-flow algorithms for energy minimization in vision. *IEEE Trans. Pattern Anal. Mach. Intell.*, 26(9):1124–1137, 2004.

[7] Y. Boykov, O. Veksler, and R. Zabih. Fast approximate energy minimization via graph cuts. *IEEE Trans. Pattern Anal. Mach. Intell.*, 23(11):1222–1239, 2001.

[8] X. Chen, A. Jain, A. Gupta, and L. Davis. Piecing together the segmentation jigsaw using context. In *CVPR*, 2011.

[9] X. Feng, C. K. I. Williams, and S. N. Felderhof. Combining belief networks and neural networks for scene segmentation. *IEEE Trans. Pattern Anal. Mach. Intell.*, 24(4):467–483, 2002.

[10] B. Fulkerson, A. Vedaldi, and S. Soatto. Class segmentation and object localization with superpixel neighborhoods. In *ICCV*, pages 670–677, 2009.

[11] S. Gould, R. Fulton, and D. Koller. Decomposing a scene into geometric and semantically consistent regions. In *ICCV*, pages 1–8, 2009.

[12] D. M. Greig, B. T. Porteous, and A. H. Seheult. Exact maximum a posteriori estimation for binary images. *Journal of the Royal Statistical Society*, 51(2), 1989.

[13] C. Gu, J. J. Lim, P. Arbelaez, and J. Malik. Recognition using regions. In *CVPR*, pages 1030–1037, 2009.

[14] T. Joachims, T. Finley, and C.-N. J. Yu. Cutting-plane training of structural SVMs. *Machine Learning*, 77(1), 2009.

[15] V. Kolmogorov, Y. Boykov, and C. Rother. Applications of parametric maxflow in computer vision. In *ICCV*, pages 1–8, 2007.

[16] V. Kolmogorov and R. Zabih. What energy functions can be minimized via graph cuts? *IEEE Trans. Pattern Anal. Mach. Intell.*, 26(2):147–159, 2004.

[17] A. Kulesza and F. Pereira. Structured learning with approximate inference. In *NIPS*, 2007.

[18] M. P. Kumar and D. Koller. Efficiently selecting regions for scene understanding. In *CVPR*, 2010.

[19] L. Ladicky, C. Russell, P. Kohli, and P. H. S. Torr. Associative hierarchical crfs for object class image segmentation. In *ICCV*, pages 739–746, 2009.

[20] T. Malisiewicz and A. A. Efros. Improving spatial support for objects via multiple segmentations. In *BMVC*, September 2007.

[21] M. Marszalek and C. Schmid. Accurate object localization with shape masks. In *CVPR*, 2007.

[22] D. Munoz, J. A. Bagnell, and M. Hebert. Stacked hierarchical labeling. In *ECCV (6)*, pages 57–70, 2010.

[23] A. Opelt, A. Pinz, M. Fussenegger, and P. Auer. Generic object recognition with boosting. *IEEE Trans. Pattern Anal. Mach. Intell.*, 28(3):416–431, 2006.

[24] N. Plath, M. Toussaint, and S. Nakajima. Multi-class image segmentation using conditional random fields and global classification. In *ICML*, page 103, 2009.

[25] J. Reynolds and K. Murphy. Figure-ground segmentation using a hierarchical conditional random field. In *CRV*, pages 175–182, 2007.

[26] P. Schnitzspan, M. Fritz, and B. Schiele. Hierarchical support vector random fields: Joint training to combine local and global features. In *ECCV (2)*, pages 527–540, 2008.

[27] E. Sharon, A. Brandt, and R. Basri. Fast multiscale image segmentation. In *CVPR*, 2000.

[28] J. Shi and J. Malik. Normalized cuts and image segmentation. In *CVPR*, pages 731–737, 1997.

[29] J. Shotton, J. M. Winn, C. Rother, and A. Criminisi. *TextonBoost*: Joint appearance, shape and context modeling for multi-class object recognition and segmentation. In *ECCV (1)*, pages 1–15, 2006.

[30] D. Singaraju and R. Vidal. Using global bag of features models in random fields for joint categorization and segmentation of objects. In *CVPR*, 2011.

[31] R. Szeliski, R. Zabih, D. Scharstein, O. Veksler, V. Kolmogorov, A. Agarwala, M. F. Tappen, and C. Rother. A comparative study of energy minimization methods for markov random fields with smoothness-based priors. *IEEE Trans. Pattern Anal. Mach. Intell.*, 30(6):1068–1080, 2008.

[32] M. Szummer, P. Kohli, and D. Hoiem. Learning crfs using graph cuts. In *ECCV*, 2008.

[33] B. Taskar, C. Guestrin, and D. Koller. Max-margin markov networks. In *NIPS*, 2003.

[34] S. Todorovic and N. Ahuja. Learning subcategory relevances for category recognition. In *CVPR*, 2008.

[35] I. Tsochantaridis, T. Hofmann, T. Joachims, and Y. Altun. Support vector machine learning for interdependent and structured output spaces. In *ICML*, 2004.

[36] A. Vedaldi and B. Fulkerson. VLFeat: An open and portable library of computer vision algorithms. http://www.vlfeat.org/, 2008.

[37] A. Vedaldi and A. Zisserman. Efficient additive kernels via explicit feature maps. In *CVPR*, 2010.

[38] O. Veksler. Image segmentation by nested cuts. In *CVPR*, pages 1339–, 2000.

[39] J. Wang, J. Yang, K. Yu, F. Lv, T. S. Huang, and Y. Gong. Locality-constrained linear coding for image classification. In *CVPR*, pages 3360–3367, 2010.

